# An Alternative Infinite Mixture Of Gaussian Process Experts

**Edward Meeds and Simon Osindero**
Department of Computer Science
University of Toronto
Toronto, M5S 3G4
{ewm,osindero}@cs.toronto.edu

## Abstract

We present an infinite mixture model in which each component comprises a multivariate Gaussian distribution over an input space, and a Gaussian Process model over an output space. Our model is neatly able to deal with non-stationary covariance functions, discontinuities, multi-modality and overlapping output signals. The work is similar to that by Rasmussen and Ghahramani [1]; however, we use a full generative model over input and output space rather than just a conditional model. This allows us to deal with incomplete data, to perform inference over inverse functional mappings as well as for regression, and also leads to a more powerful and consistent Bayesian specification of the effective 'gating network' for the different experts.

## 1 Introduction

Gaussian process (GP) models are powerful tools for regression, function approximation, and predictive density estimation. However, despite their power and flexibility, they suffer from several limitations. The computational requirements scale cubically with the number of data points, thereby necessitating a range of approximations for large datasets. Another problem is that it can be difficult to specify priors and perform learning in GP models if we require non-stationary covariance functions, multi-modal output, or discontinuities.

There have been several attempts to circumvent some of these lacunae, for example [2, 1]. In particular the Infinite Mixture of Gaussian Process Experts (IMoGPE) model proposed by Rasmussen and Ghahramani [1] neatly addresses the aforementioned key issues. In a single GP model, an $n$ by $n$ matrix must be inverted during inference. However, if we use a model composed of multiple GP's, each responsible only for a subset of the data, then the computational complexity of inverting an $n$ by $n$ matrix is replaced by several inversions of smaller matrices — for large datasets this can result in a substantial speed-up and may allow one to consider large-scale problems that would otherwise be unwieldy. Furthermore, by combining multiple stationary GP experts, we can easily accommodate non-stationary covariance and noise levels, as well as distinctly multi-modal outputs. Finally, by placing a Dirichlet process prior over the experts we can allow the data and our prior beliefs (which may be rather vague) to automatically determine the number of components to use.

In this work we present an alternative infinite model that is strongly inspired by the work in [1], but which uses a different formulation for the mixture of experts that is in the style presented in, for example [3, 4]. This alternative approach effectively uses posterior re-

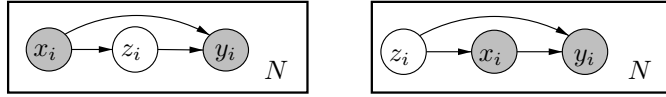

Figure 1: **Left:** Graphical model for the standard MoE model [6]. The expert indicators $\{z_{(i)}\}$ are specified by a gating network applied to the inputs $\{\mathbf{x}_{(i)}\}$. **Right:** An alternative view of MoE model using a full generative model [4]. The distribution of input locations is now given by a mixture model, with components for each expert. Conditioned on the input locations, the posterior responsibilities for each mixture component behave like a gating network.

sponsibilities from a mixture distribution as the gating network. Even if the task at hand is simply output density estimation or regression, we suggest a full generative model over inputs and outputs might be preferable to a purely conditional model. The generative approach retains all the strengths of [1] and also has a number of potential advantages, such as being able to deal with partially specified data (e.g. missing input co-ordinates) and being able to infer inverse functional mappings (i.e. the input space given an output value). The generative approach also affords us a richer and more consistent way of specifying our prior beliefs about how the covariance structure of the outputs might vary as we move within input space.

An example of the type of generative model which we propose is shown in figure 2. We use a Dirichlet process prior over a countably infinite number of experts and each expert comprises two parts: a density over input space describing the distribution of input points associated with that expert, and a Gaussian Process model over the outputs associated with that expert. In this preliminary exposition, we restrict our attention to experts whose input space densities are given a single full covariance Gaussian. Even this simple approach demonstrates interesting performance and capabilities. However, in a more elaborate setup the input density associated with each expert might itself be an infinite mixture of simpler distributions (for instance, an infinite mixture of Gaussians [5]) to allow for the most flexible partitioning of input space amongst the experts.

The structure of the paper is as follows. We begin in section 2 with a brief overview of two ways of thinking about Mixtures of Experts. Then, in section 3, we give the complete specification and graphical depiction of our generative model, and in section 4 we outline the steps required to perform Monte Carlo inference and prediction. In section 5 we present the results of several simple simulations that highlight some of the salient features of our proposal, and finally in section 6, we discuss our work and place it in relation to similar techniques.

## 2   Mixtures of Experts

In the standard mixture of experts (MoE) model [6], a gating network probabilistically mixes regression components. One subtlety in using GP's in a mixture of experts model is that IID assumptions on the data no longer hold and we must specify joint distributions for each possible assignment of experts to data. Let $\{\mathbf{x}_{(i)}\}$ be the set of $d$-dimensional input vectors, $\{y_{(i)}\}$ be the set of scalar outputs, and $\{z_{(i)}\}$ be the set of expert indicators which assign data points to experts.

The likelihood of the outputs, given the inputs, is specified in equation 1, where $\theta_r^{\mathrm{GP}}$ represents the GP parameters of the $r$th expert, $\theta^g$ represents the parameters of the gating network, and the summation is over all possible configurations of indicator variables.

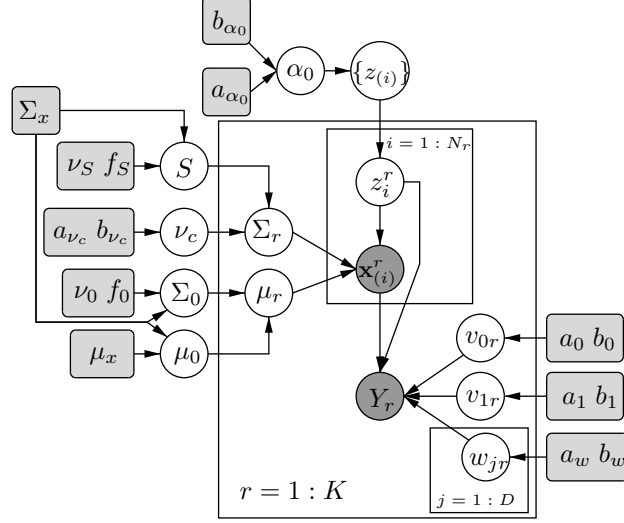

Figure 2: The graphical model representation of the alternative infinite mixture of GP experts (AiMoGPE) model proposed in this paper. We have used $\mathbf{x}_{(i)}^r$ to represent the $i$th data point in the set of input data whose expert label is $r$, and $Y_r$ to represent the set of all output data whose expert label is $r$. In other words, input data are IID given their expert label, whereas the *sets* of output data are IID given their corresponding *sets* of input data. The lightly shaded boxes with rounded corners represent hyper-hyper parameters that are fixed ($\Omega$ in the text). The DP concentration parameter $\alpha_0$, the expert indicators variables, $\{z_{(i)}\}$, the gate hyperparameters, $\phi^{\mathbf{x}} = \{\mu_0, \Sigma_0, \nu_c, S\}$, the gate component parameters, $\psi_r^{\mathbf{x}} = \{\mu_r, \Sigma_r\}$, and the GP expert parameters, $\theta_r^{\mathrm{GP}} = \{v_{0r}, v_{1r}, w_{jr}\}$, are all updated for all $r$ and $j$.

$$P(\{y_{(i)}\}|\{\mathbf{x}_{(i)}\}, \theta) = \sum_{\mathcal{Z}} P(\{z_{(i)}\}|\{\mathbf{x}_{(i)}\}, \theta^g) \prod_r P(\{y_{(i)} : z_{(i)} = r\}|\{\mathbf{x}_{(i)} : z_{(i)} = r\}, \theta_r^{\mathrm{GP}})$$

(1)

There is an alternative view of the MoE model in which the experts also generate the inputs, rather than simply being conditioned on them [3, 4] (see figure 1). This alternative view employs a joint mixture model over input and output space, even though the objective is still primarily that of estimating conditional densities i.e. outputs given inputs. The gating network effectively gets specified by the posterior responsibilities of each of the different components in the mixture. An advantage of this perspective is that it can easily accommodate partially observed inputs and it also allows 'reverse-conditioning', should we wish to estimate where in input space a given output value is likely to have originated. For a mixture model using Gaussian Processes experts, the likelihood is given by

$$P(\{\mathbf{x}_{(i)}\}, \{y_{(i)}\}|\theta) = \sum_{\mathcal{Z}} P(\{z_{(i)}\}|\theta^g) \times$$

$$\prod_r P(\{y_{(i)} : z_{(i)} = r\}|\{\mathbf{x}_{(i)} : z_{(i)} = r\}, \theta_r^{\mathrm{GP}}) P(\{\mathbf{x}_{(i)} : z_{(i)} = r\}|\theta^g) \quad (2)$$

where the description of the density over input space is encapsulated in $\theta^g$.

## 3 Infinite Mixture of Gaussian Processes: A Joint Generative Model

The graphical structure for our full generative model is shown in figure 2. Our generative process does not produce IID data points and is therefore most simply formulated either as

a joint distribution over a dataset of a given size, or as a set of conditionals in which we incrementally add data points.To construct a complete set of $N$ sample points from the prior (specified by top-level hyper-parameters $\Omega$) we would perform the following operations:

1. Sample Dirichlet process concentration variable $\alpha_0$ given the top-level hyper-parameters.
2. Construct a partition of $N$ objects into at most $N$ groups using a Dirichlet process. This assignment of objects is denoted by using a set the indicator variables $\{z_{(i)}\}_{i=1}^N$.
3. Sample the gate hyperparameters $\phi^{\mathbf{x}}$ given the top-level hyperparameters.
4. For each grouping of indicators $\{z_{(i)} : z_{(i)} = r\}$, sample the input space parameters $\psi_r^{\mathbf{x}}$ conditioned on $\phi^{\mathbf{x}}$. $\psi_r^{\mathbf{x}}$ defines the density in input space, in our case a full-covariance Gaussian.
5. Given the parameters $\psi_r^{\mathbf{x}}$ for each group, sample the locations of the input points $X_r \equiv \{\mathbf{x}_{(i)} : z_{(i)} = r\}$.
6. For each group, sample the hyper-parameters for the GP expert associated with that group, $\theta_r^{\mathrm{GP}}$.
7. Using the input locations $X_r$ and hyper-parameters $\theta_r^{\mathrm{GP}}$ for the individual groups, formulate the GP output covariance matrix and sample the set of output values, $Y_r \equiv \{y_{(i)} : z_{(i)} = r\}$ from this joint Gaussian distribution.

We write the full joint distribution of our model as follows.

$$P(\{\mathbf{x}_{(i)}, y_{(i)}\}_{i=1}^N, \{z_{(i)}\}_{i=1}^N, \{\psi_r^{\mathbf{x}}\}_{r=1}^N, \{\theta_r^{\mathrm{GP}}\}_{r=1}^N, \alpha_0, \phi^{\mathbf{x}}|N, \Omega) =$$
$$\prod_{r=1}^N \left[ H_r^N P(\psi_r^{\mathbf{x}}|\phi^{\mathbf{x}}) P(X_r|\psi_r^{\mathbf{x}}) P(\theta_r^{\mathrm{GP}}|\Omega) P(Y_r|X_r, \theta_r^{\mathrm{GP}}) + (1 - H_r^N) D_0(\psi_r^{\mathbf{x}}, \theta_r^{\mathrm{GP}}) \right]$$
$$\times P(\{z_{(i)}\}_{i=1}^N|N, \alpha_0) P(\alpha_0|\Omega) P(\phi^{\mathbf{x}}|\Omega) \tag{3}$$

Where we have used the supplementary notation: $H_r^N = 0$ if $\{\{z_{(i)}\} : z_{(i)} = r\}$ is the empty set and $H_r^N = 1$ otherwise; and $D_0(\psi_r^{\mathbf{x}}, \theta_r^{\mathrm{GP}})$ is a delta function on an (irrelevant) dummy set of parameters to ensure proper normalisation.

For the GP components, we use a standard, stationary covariance function of the form

$$Q(\mathbf{x}_{(i)}, \mathbf{x}_{(h)}) = v_0 \exp\left(-\frac{1}{2}\sum_{j=1}^D \left(x_{(i)j} - x_{(h)j}\right)^2 / w_j^2\right) + \delta(i, h)v_1 \tag{4}$$

The individual distributions in equation 3 are defined as follows[1]:

$$P(\alpha_0|\Omega) = \mathcal{G}(\alpha_0; \, a_{\alpha_0}, b_{\alpha_0}) \tag{5}$$
$$P(\{z_{(i)}\}_{i=1}^N|N, \Omega) = \mathcal{PU}(\alpha_0, N) \tag{6}$$
$$P(\phi^{\mathbf{x}}|\Omega) = \mathcal{N}(\mu_0; \, \mu_x, \Sigma_x/f_0)\mathcal{W}(\Sigma_0^{-1}; \, \nu_0, f_0\Sigma_x^{-1}/\nu_0)$$
$$\mathcal{G}(\nu_c; \, a_{\nu_c}, b_{\nu_c})\mathcal{W}(S^{-1}; \, \nu_S, f_S\Sigma_x/\nu_S) \tag{7}$$
$$P(\psi_r^{\mathbf{x}}|\Omega) = \mathcal{N}(\mu_r; \, \mu_0, \Sigma_0)\mathcal{W}(\Sigma_r^{-1}; \, \nu_c, S/\nu_c) \tag{8}$$
$$P(X_r|\psi_r^{\mathbf{x}}) = \mathcal{N}(X_r; \, \mu_r, \Sigma_r) \tag{9}$$
$$P(\theta_r^{\mathrm{GP}}|\Omega) = \mathcal{G}(v_{0r}; \, a_0, b_0)\mathcal{G}(v_{1r}; \, a_1, b_1)\prod_{j=1}^D \mathcal{LN}(w_{jr}; \, a_w, b_w) \tag{10}$$
$$P(Y_r|X_r, \theta_r^{\mathrm{GP}}) = \mathcal{N}(Y_r; \, \mu_{Q_r}, \sigma_{Q_r}^2) \tag{11}$$

In an approach similar to Rasmussen [5], we use the input data mean $\mu_x$ and covariance $\Sigma_x$ to provide an automatic normalisation of our dataset. We also incorporate additional hyperparameters $f_0$ and $f_S$, which allow prior beliefs about the variation in location of $\mu_r$ and size of $\Sigma_r$, relative to the data covariance.

## 4 Monte Carlo Updates

Almost all the integrals and summations required for inference and learning operations within our model are analytically intractable, and therefore necessitate Monte Carlo approximations. Fortunately, all the necessary updates are relatively straightforward to carry out using a Markov Chain Monte Carlo (MCMC) scheme employing Gibbs sampling and Hybrid Monte Carlo. We also note that in our model the predictive density depends on the entire set of test locations (in input space). This transductive behaviour follows from the non-IID nature of the model and the influence that test locations have on the posterior distribution over mixture parameters. Consequently, the marginal predictive distribution at a given location can depend on the other locations for which we are making simultaneous predictions. This may or may not be desired. In some situations the ability to incorporate the additional information about the input density at test time may be beneficial. However, it is also straightforward to effectively 'ignore' this new information and simply compute a set of independent single location predictions.

Given a set of test locations $\{\mathbf{x}_{(t)}^*\}$, along with training data pairs $\{\mathbf{x}_{(i)}, y_{(i)}\}$ and top-level hyper-parameters $\Omega$, we iterate through the following conditional updates to produce our predictive distribution for unknown outputs $\{y_{(t)}^*\}$. The parameter updates are all conjugate with the prior distributions, except where noted:

1. Update indicators $\{z_{(i)}\}$ by cycling through the data and sampling one indicator variable at a time. We use algorithm 8 from [9] with $m = 1$ to explore new experts.
2. Update input space parameters.
3. Update GP hyper-params using Hybrid Monte Carlo [10].
4. Update gate hyperparameters. Note that $\nu_c$ is updated using slice sampling [11].
5. Update DP hyperparameter $\alpha_0$ using the data augmentation technique of Escobar and West [12].
6. Resample missing output values by cycling through the experts, and jointly sampling the missing outputs associated with that GP.

We perform some preliminary runs to estimate the longest auto-covariance time, $\tau_{\max}$ for our posterior estimates, and then use a burn-in period that is about 10 times this timescale before taking samples every $\tau_{\max}$ iterations.[2] For our simulations the auto-covariance time was typically 40 complete update cycles, so we use a burn-in period of 500 iterations and collect samples every 50.

## 5 Experiments

### 5.1 Samples From The Prior

In figure 3 (A) we give an example of data drawn from our model which is multi-modal and non-stationary. We also use this artificial dataset to confirm that our MCMC algorithm performs well and is able recover sensible posterior distributions. Posterior histograms for some of the inferred parameters are shown in figure 3 (B) and we see that they are well clustered around the 'true' values.

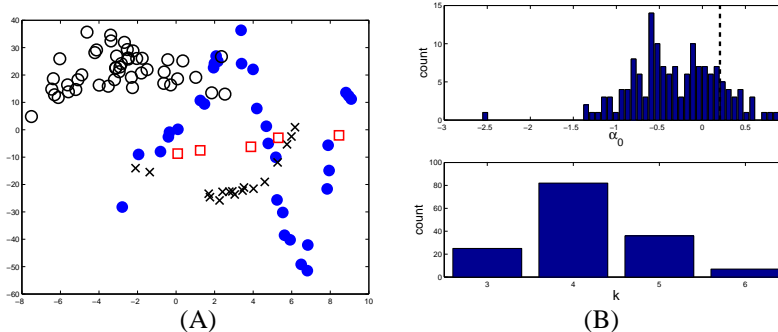

(A)                                                  (B)

Figure 3: (A) A set of samples from our model prior. The different marker styles are used to indicate the sets of points from different experts. (B) The posterior distribution of $\log \alpha_0$ with its true value indicated by the dashed line (top) and the distribution of occupied experts (bottom). We note that the posterior mass is located in the vicinity of the true values.

## 5.2 Inference On Toy Data

To illustrate some of the features of our model we constructed a toy dataset consisting of $4$ continuous functions, to which we added different levels of noise. The functions used were:

$$
\begin{aligned}
f_1(a_1) &= 0.25a_1^2 - 40 & a_1 &\in (0 \ldots 15) & \text{Noise SD: } 7 & \quad(12)\\
f_2(a_2) &= -0.0625(a_2 - 18)^2 + .5a_2 + 20 & a_2 &\in (35 \ldots 60) & \text{Noise SD: } 7 & \quad(13)\\
f_3(a_3) &= 0.008(a_3 - 60)^3 - 70 & a_3 &\in (45 \ldots 80) & \text{Noise SD: } 4 & \quad(14)\\
f_4(a_4) &= -\sin(0.25a_4) - 6 & a_4 &\in (80 \ldots 100) & \text{Noise SD: } 2 & \quad(15)
\end{aligned}
$$

The resulting data has non-stationary noise levels, non-stationary covariance, discontinuities and significant multi-modality. Figure 4 shows our results on this dataset along with those from a single GP for comparison.

We see that in order to account for the entire data set with a single GP, we are forced to infer an unnecessarily high level of noise in the function. Also, a single GP is unable to capture the multi-modality or non-stationarity of the data distribution. In contrast, our model seems much more able to deal with these challenges.

Since we have a full generative model over both input and output space, we are also able to use our model to infer likely input locations given a particular output value. There are a number of applications for which this might be relevant, for example if one wanted to sample candidate locations at which to evaluate a function we are trying to optimise. We provide a simple illustration of this in figure 4 (B). We choose three output levels and conditioned on the output having these values, we sample for the input location. The inference seems plausible and our model is able to suggest locations in input space for a maximal output value ($+40$) that was not seen in the training data.

## 5.3 Regression on a simple "real-world" dataset

We also apply our model and algorithm to the motorcycle dataset of [13]. This is a commonly used dataset in the GP community and therefore serves as a useful basis for comparison. In particular, it also makes it easy to see how our model compares with standard GP's and with the work of [1]. Figure 5 compares the performance of our model with that of a single GP. In particular, we note that although the median of our model closely resembles the mean of the single GP, our model is able to more accurately model the low noise level

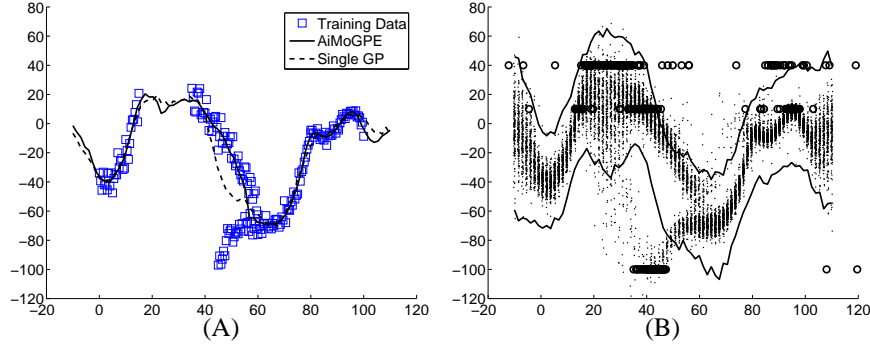

Figure 4: Results on a toy dataset. (A) The training data is shown along with the predictive mean of a stationary covariance GP and the median of the predictive distribution of our model. (B) The small dots are samples from the model (160 samples per location) evaluated at 80 equally spaced locations across the range (but plotted with a small amount of jitter to aid visualisation). These illustrate the predictive density from our model. The solid the lines show the $\pm\,2$ SD interval from a regular GP. The circular markers at ordinates of $40$, $10$ and $-100$ show samples from 'reverse-conditioning' where we sample likely abscissa locations given the test ordinate and the set of training data.

on the left side of the dataset. For the remainder of the dataset, the noise level modeled by our model and a single GP are very similar, although our model is better able to capture the behaviour of the data at around 30 ms. It is difficult to make an exact comparison to [1], however we can speculate that our model is more realistically modeling the noise at the beginning of the dataset by not inferring an overly "flat" GP expert at that location. We can also report that our expert adjacency matrix closely resembles that of [1].

## 6   Discussion

We have presented an alternative framework for an infinite mixture of GP experts. We feel that our proposed model carries over the strengths of [1] and augments these with the several desirable additional features. The pseudo-likelihood objective function used to adapt the gating network defined in [1] is not guaranteed to lead to a self-consistent distribution and therefore the results may depend on the order in which the updates are performed; our model incorporates a consistent Bayesian density formulation for both input and output spaces by definition. Furthermore, in our most general framework we are more naturally able to specify priors over the partitioning of space between different expert components. Also, since we have a full joint model we can infer inverse functional mappings.

There should be considerable gains to be made by allowing the input density models be more powerful. This would make it easier for arbitrary regions of space to share the same covariance structures; at present the areas 'controlled' by a particular expert tend to be local. Consequently, a potentially undesirable aspect of the current model is that strong clustering in input space can lead us to infer several expert components even if a single GP would do a good job of modelling the data. An elegant way of extending the model in this way might be to use a separate infinite mixture distribution for the input density of *each* expert, perhaps incorporating a hierarchical DP prior across the infinite set of experts to allow information to be shared.

With regard to applications, it might be interesting to further explore our model's capability to infer inverse functional mappings; perhaps this could be useful in an optimisation or active learning context. Finally, we note that although we have focused on rather small examples so far, it seems that the inference techniques should scale well to larger problems

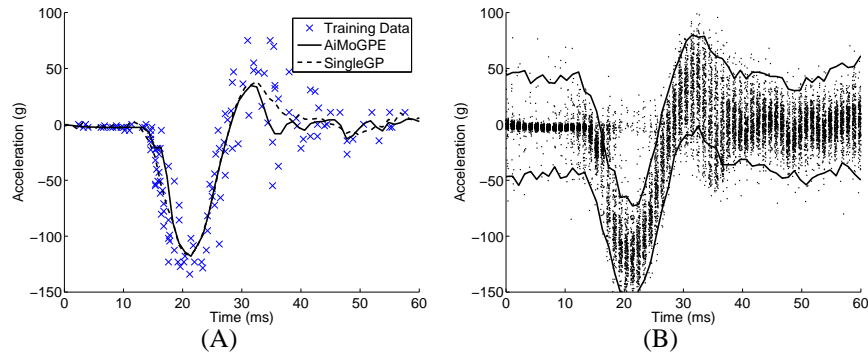

Figure 5: (A) Motorcycle impact data together with the median of our model's point-wise predictive distribution and the predictive mean of a stationary covariance GP model. (B) The small dots are samples from our model (160 samples per location) evaluated at 80 equally spaced locations across the range (but plotted with a small amount of jitter to aid visualisation). The solid lines show the ± 2 SD interval from a regular GP.

and more practical tasks.

### Acknowledgments

Thanks to Ben Marlin for sharing slice sampling code and to Carl Rasmussen for making `minimize.m` available.

## Footnotes

[1]We use the notation $\mathcal{N}, \mathcal{W}, \mathcal{G}$, and $\mathcal{LN}$ to represent the normal, the Wishart, the gamma, and the log-normal distributions, respectively; we use the parameterizations found in [7] (Appendix A). The notation $\mathcal{PU}$ refers to the Polya urn distribution [8].

[2]This is primarily for convenience. It would also be valid to use all the samples after the burn-in period, and although they could not be considered independent, they could be used to obtain a more accurate estimator.

## References

[1] C.E. Rasmussen and Z. Ghahramani. Infinite mixtures of Gaussian process experts. In *Advances in Neural Information Processing Systems 14*, pages 881–888. MIT Press, 2002.

[2] V. Tresp. Mixture of Gaussian processes. In *Advances in Neural Information Processing Systems*, volume 13. MIT Press, 2001.

[3] Z. Ghahramani and M. I. Jordan. Supervised learning from incomplete data via an EM approach. In *Advances in Neural Information Processing Systems 6*, pages 120–127. Morgan-Kaufmann, 1995.

[4] L. Xu, M. I. Jordan, and G. E. Hinton. An alternative model for mixtures of experts. In *Advances in Neural Information Processing Systems 7*, pages 633–640. MIT Press, 1995.

[5] C. E. Rasmussen. The infinite Gaussian mixture model. In *Advances in Neural Information Processing Systems*, volume 12, pages 554–560. MIT Press, 2000.

[6] R.A. Jacobs, M.I. Jordan, and G.E. Hinton. Adaptive mixture of local experts. *Neural Computation*, 3, 1991.

[7] A. Gelman, J. B. Carlin, H. S. Stern, and D. B. Rubin. *Bayesian Data Analysis*. Chapman and Hall, 2nd edition, 2004.

[8] D. Blackwell and J. B. MacQueen. Ferguson distributions via Polya urn schemes. *The Annals of Statistics*, 1(2):353–355, 1973.

[9] R. M. Neal. Markov chain sampling methods for Dirichlet process mixture models. *Journal of Computational and Graphical Statistics*, 9:249–265, 2000.

[10] R. M. Neal. Probabilistic inference using Markov chain Monte Carlo methods. Technical Report CRG-TR-93-1, University of Toronto, 1993.

[11] R. M. Neal. Slice sampling (with discussion). *Annals of Statistics*, 31:705–767, 2003.

[12] M. Escobar and M. West. Computing Bayesian nonparametric hierarchical models. In *Practical Nonparametric and Semiparametric Bayesian Statistics*, number 133 in Lecture Notes in Statistics. Springer-Verlag, 1998.

[13] B. W. Silverman. Some aspects of the spline smoothing approach to non-parametric regression curve fitting. *J. Royal Stayt Society. Ser. B*, 47:1–52, 1985.